# Binet-Cauchy Kernels

**S.V.N. Vishwanathan, Alexander J. Smola**
National ICT Australia, Machine Learning Program, Canberra, ACT 0200, Australia
{SVN.Vishwanathan, Alex.Smola}@nicta.com.au

## Abstract

We propose a family of kernels based on the Binet-Cauchy theorem and its extension to Fredholm operators. This includes as special cases all currently known kernels derived from the behavioral framework, diffusion processes, marginalized kernels, kernels on graphs, and the kernels on sets arising from the subspace angle approach. Many of these kernels can be seen as the extrema of a new continuum of kernel functions, which leads to numerous new special cases. As an application, we apply the new class of kernels to the problem of clustering of video sequences with encouraging results.

## 1 Introduction

Recent years have see a combinatorial explosion of results on kernels for structured and semi-structured data, including trees, strings, graphs, transducers and dynamical systems [6, 8, 15, 13]. The fact that these kernels are very specific to the type of discrete data under consideration is a major cause of confusion to the practitioner. What is required is a) an *unified* view of the field and b) a *recipe* to design new kernels easily.

The present paper takes a step in this direction by unifying these diverse kernels by means of the Binet-Cauchy theorem. Our point of departure is the work of Wolf and Shashua [17], or more specifically, their proof that $\det A^\top B$ is a kernel on matrices $A, B \in \mathbb{R}^{m \times n}$. We extend the results of [17] in the following three ways:

1. There exists an operator-valued equivalent of the Binet-Cauchy theorem.
2. Wolf and Shashua only exploit the Binet-Cauchy theorem for one particular choice of parameters. It turns out that the continuum of these values corresponds to a large class of kernels some of which are well known and others which are novel.
3. The Binet-Cauchy theorem can be extended to semirings. This points to a close connection with rational kernels [3].

**Outline of the paper:** Section 2 contains the main result of the present paper: the definition of Binet-Cauchy kernels and their efficient computation. Subsequently, section 3 discusses a number of special cases, which allows us to recover well known kernel functions. Section 4 applies our derivations to the analysis of video sequences, and we conclude with a discussion of our results.

## 2 Binet-Cauchy Kernels

In this section we deal with linear mappings from $\mathcal{X} = \mathbb{R}^n$ to $\mathcal{Y} = \mathbb{R}^m$ (typically denoted by matrices), their coordinate free extensions to Fredholm operators (here $\mathbb{R}^n$ and $\mathbb{R}^m$ are replaced by measurable sets), and their extensions to semirings (here addition and multiplication are replaced by an abstract class of symbols $(\oplus, \otimes)$ with the same distributive properties).

## 2.1 The General Composition Formula

We begin by defining compound matrices. They arise by picking subsets of entries of a matrix and computing their determinants.

**Definition 1 (Compound Matrix)** *Let $A \in \mathbb{R}^{m \times n}$, let $q \leq \min(m, n)$ and let $I_q^n = \{\mathbf{i} = (i_1, i_2, \ldots, i_q) : 1 \leq i_1 < \ldots < i_q \leq n, i_i \in \mathbb{N}\}$ and likewise $I_q^m$. Then the compound matrix of order $q$ is defined as*

$$[C_q(A)]_{\mathbf{i},\mathbf{j}} := \det(A(i_k, j_l))_{k,l=1}^q \text{ where } \mathbf{i} \in I_q^n \text{ and } \mathbf{j} \in I_q^m. \tag{1}$$

*Here $\mathbf{i}, \mathbf{j}$ are assumed to be arranged in lexicographical order.*

**Theorem 2 (Binet-Cauchy)** *Let $A \in \mathbb{R}^{l \times m}$ and, $B \in \mathbb{R}^{l \times n}$. For $q \leq \min(m, n, l)$ we have $C_q(A^\top B) = C_q(A)^\top C_q(B)$.*

When $q = m = n = l$ we have $C_q(A) = \det(A)$ and the Binet-Cauchy theorem becomes the well known identity $\det(A^\top B) = \det(A) \det(B)$. On the other hand when $q = 1$ we have $C_1(A) = A$, so Theorem 2 reduces to a tautology.

**Theorem 3 (Binet-Cauchy for Semirings)** *When the common semiring $(\mathbb{R}, +, \cdot, 0, 1)$ is replaced by an abstract semiring $(\mathbb{K}, \oplus, \otimes, \bar{0}, \bar{1})$ the equality $C_q(A^\top B) = C_q(A)^\top C_q(B)$ still holds. Here all operations occur on the monoid $\mathbb{K}$, addition and multiplication are replaced by $\oplus, \otimes$, and $(\bar{0}, \bar{1})$ take the role of $(0, 1)$.*

A second extension of Theorem 2 is to replace matrices by Fredholm operators, as they can be expressed as integral operators with corresponding kernels. In this case, Theorem 2 becomes a statement about convolutions of integral kernels.

**Definition 4 (Fredholm Operator)** *A Fredholm operator is a bounded linear operator between two Hilbert spaces with closed range and whose kernel and co-kernel are finite-dimensional.*

**Theorem 5 (Kernel Representation of Fredholm Operators)** *Let $A : L_2(\mathcal{Y}) \to L_2(\mathcal{X})$ and, $B : L_2(\mathcal{Y}) \to L_2(\mathcal{Z})$ be Fredholm operators. Then there exists some $k_A : \mathcal{X} \times \mathcal{Y} \to \mathbb{R}$ such that for all $f \in L_2(\mathcal{X})$ we have*

$$[Af](x) = \int_{\mathcal{Y}} k_A(x, y) f(y) dy. \tag{2}$$

*Moreover, for the composition $A^\top B$ we have $k_{A^\top B}(x, z) = \int_{\mathcal{Y}} k_{A^\top}(x, y) k_B(y, z) dy$.*

Here the convolution of kernels $k_A$ and $k_B$ plays the same role as the matrix multiplication. To extend the Binet-Cauchy theorem we need to introduce the analog of compound matrices:

**Definition 6 (Compound Kernel and Operator)** *Denote by $\mathcal{X}, \mathcal{Y}$ ordered sets and let $k : \mathcal{X} \times \mathcal{Y} \to \mathbb{R}$. Define $I_q^{\mathcal{X}} = \{\mathbf{x} \in \mathcal{X}^q : x_1 \leq \ldots \leq x_q\}$ and likewise $I_q^{\mathcal{Y}}$. Then the compound kernel of order $q$ is defined as*

$$k^{[q]}(\mathbf{x}, \mathbf{y}) := \det(k(x_k, y_l))_{k,l=1}^q \text{ where } \mathbf{x} \in I_q^{\mathcal{X}} \text{ and } \mathbf{y} \in I_q^{\mathcal{Y}}. \tag{3}$$

*If $k$ is the integral kernel of an operator $A$ we define $C_q(A)$ to be the integral operator corresponding to $k^{[q]}$.*

**Theorem 7 (General Composition Formula [11])** *Let $\mathcal{X}, \mathcal{Y}, \mathcal{Z}$ be ordered sets and let $A : L_2(\mathcal{Y}) \to L_2(\mathcal{X})$, $B : L_2(\mathcal{Y}) \to L_2(\mathcal{Z})$ be Fredholm operators. Then for $q \in \mathbb{N}$ we have*

$$C_q(A^\top B) = C_q(A)^\top C_q(B). \tag{4}$$

To recover Theorem 2 from Theorem 7 set $\mathcal{X} = [1..m]$, $\mathcal{Y} = [1..n]$ and $\mathcal{Z} = [1..l]$.

## 2.2 Kernels

The key idea in turning the Binet-Cauchy theorem and its various incarnations into a kernel is to exploit the fact that $\operatorname{tr} A^\top B$ and $\det A^\top B$ are kernels on operators $A, B$. We extend this by replacing $A^\top B$ with some functions $\psi(A)^\top \psi(B)$ involving compound operators.

**Theorem 8 (Trace and Determinant Kernel)** *Let $A, B : L_2(\mathcal{X}) \to L_2(\mathcal{Y})$ be Fredholm operators and let $S : L_2(\mathcal{Y}) \to L_2(\mathcal{Y})$, $T : L_2(\mathcal{X}) \to L_2(\mathcal{X})$ be positive trace-class operators. Then the following two kernels are well defined and they satisfy Mercer's condition:*

$$k(A, B) = \operatorname{tr}\left[ SA^\top TB \right] \tag{5}$$

$$k(A, B) = \det\left[ SA^\top TB \right]. \tag{6}$$

Note that determinants are not defined in general for infinite dimensional operators, hence our restriction to Fredholm operators $A, B$ in (6).

**Proof** Observe that $S$ and $T$ are positive and compact. Hence they admit a decomposition into $S = V_S V_S^\top$ and $T = V_T^\top V_T$. By virtue of the commutativity of the trace we have that $k(A, B) = \operatorname{tr}\left( \left[V_T A V_S\right]^\top \left[V_T B V_S\right] \right)$. Analogously, using the Binet-Cauchy theorem, we can decompose the determinant. The remaining terms $V_T A V_S$ and $V_T B V_S$ are again Fredholm operators for which determinants are well defined. ∎

Next we use special choices of $A, B, S, T$ involving compound operators directly to state the main theorem of our paper.

**Theorem 9 (Binet-Cauchy Kernel)** *Under the assumptions of Theorem 8 it follows that for all $q \in \mathbb{N}$ the kernels $k(A, B) = \operatorname{tr} C_q\left[SA^\top TB\right]$ and $k(A, B) = \det C_q\left[SA^\top TB\right]$ satisfy Mercer's condition.*

**Proof** We exploit the factorization $S = V_S V_S^\top, T = V_T^\top V_T$ and apply Theorem 7. This yields $C_q(SA^\top TB) = C_q(V_T A V_S)^\top C_q(V_T B V_S)$, which proves the theorem. ∎

Finally, we define a kernel based on the Fredholm determinant itself. It is essentially a weighted combination of Binet-Cauchy kernels. Fredholm determinants are defined as follows [11]:

$$D(A, \mu) := \sum_{q=1}^{\infty} \frac{\mu^q}{q!} \operatorname{tr} C_q(A). \tag{7}$$

This series converges for all $\mu \in \mathbb{C}$ and it is an entire function of $\mu$. It suggests a kernel involving weighted combinations of the kernels of Theorem 9. We have the following:

**Corollary 10 (Fredholm Kernel)** *Let $A, B, S, T$ as in Theorem 9 and let $\mu > 0$. Then the following kernel satisfies Mercer's condition:*

$$k(A, B) := D(A^\top B, \mu) \text{ where } \mu > 0. \tag{8}$$

$D(A^\top B, \mu)$ is a weighted combination of the kernels discussed above. The exponential down-weighting via $\frac{1}{q!}$ ensures that the series converges even in the case of exponential growth of the values of the compound kernel.

## 2.3 Efficient Computation

At first glance, computing the kernels of Theorem 9 and Corollary 10 presents a formidable computational task even in the finite dimensional case. If $A, B \in \mathbb{R}^{m \times n}$, the matrix $C_q(A^\top B)$ has $\binom{n}{q}$ rows and columns and each of the entries requires the computation of a determinant of a $q$-dimensional matrix. A brute-force approach would involve $O(q^3 n^q)$ operations (assuming $2q \leq n$). Clearly we need more efficient techniques.

When computing determinants, we can take recourse to Franke's Theorem [7] which states that

$$\det C_q(A) = (\det A)^{\binom{n-1}{q-1}}. \tag{9}$$

and consequently $k(A, B) = \det C_q[SA^\top TB] = (\det[SA^\top TB])^{\binom{n-1}{q-1}}$.[1] This indicates that the determinant kernel may be of limited use, due to the typically quite high power in the exponent. Kernels building on $\operatorname{tr} C_q$ are not plagued by this problem and we give an efficient recursion below. It follows from the ANOVA kernel recursion of [1]:

**Lemma 11** *Denote by $A \in \mathbb{C}^{m \times m}$ a square matrix and let $\lambda_1, \ldots, \lambda_m$ be its eigenvalues. Then $\operatorname{tr} C_q(A)$ can be computed by the following recursion:*

$$\operatorname{tr} C_q(A) = \frac{1}{q}\sum_{j=1}^{q}(-1)^{j+1}\bar{C}_{q-j}(A)\bar{C}_j(A) \text{ where } \bar{C}_q(A) = \sum_{j=1}^{n}\lambda_j^q. \qquad (10)$$

**Proof** We begin by writing $A$ in its Jordan normal form as $A = PDP^{-1}$ where $D$ is a block diagonal, upper triangular matrix. Furthermore, the diagonal elements of $D$ consist of the eigenvalues of $A$. Repeated application of the Binet-Cauchy Theorem yields

$$\operatorname{tr} C_q(A) = \operatorname{tr} C_q(P)C_q(D)C_q(P^{-1}) = \operatorname{tr} C_q(D)C_q(P^{-1})C_q(P) = \operatorname{tr} C_q(D) \quad (11)$$

For a triangular matrix the determinant is the product of its diagonal entries. Since all the square submatrices of $D$ are also upper triangular, to construct $\operatorname{tr}(C_q(D))$ we need to sum over all products of exactly $q$ eigenvalues. This is analog to the requirement of the ANOVA kernel of [1]. In its simplified version it can be written as (10), which completes the proof. ∎

We can now compute the Jordan normal form of $SA^\top TB$ in $O(n^3)$ time and apply Lemma 11 directly to it to compute the kernel value.

Finally, in the case of Fredholm determinants, we can use the recursion directly, because for $n$-dimensional matrices the sum terminates after $n$ terms. This is no more expensive than computing $\operatorname{tr} C_q$ directly. Note that in the general nonsymmetric case (i.e. $A \neq A^\top$) no such efficient recursions are known.

## 3 Special Cases

We now focus our attention on various special cases to show how they fit into the general framework which we developed in the previous section. For this to succeed, we will map various systems such as graphs, dynamical systems, or video sequences into Fredholm operators. A suitable choice of this mapping and of the operators $S, T$ of Theorem 9 will allow us to recover many well-known kernels as special cases.

### 3.1 Dynamical Systems

We begin by describing a partially observable discrete time LTI (Linear Time Invariant) model commonly used in control theory. Its time-evolution equations are given by

$$y_t = Px_t + w_t \qquad \text{where } w_t \sim \mathcal{N}(0, R) \qquad (12a)$$
$$x_t = Qx_{t-1} + v_t \qquad \text{where } v_t \sim \mathcal{N}(0, S). \qquad (12b)$$

Here $y_t \in \mathbb{R}^m$ is observed, $x_t \in \mathbb{R}^n$ is the *hidden* or *latent* variable, and $P \in \mathbb{R}^{m \times n}, Q \in \mathbb{R}^{n \times n}, R \in \mathbb{R}^{m \times m}$ and, $S \in \mathbb{R}^{n \times n}$, moreover $R, S \succeq 0$. Typically $m \gg n$. similar model exists for *continuous* LTI. Further details on it can be found in [14].

Following the behavioral framework of [16] we associate dynamical systems, $X := (P, Q, R, S, x_0)$, with their trajectories, that is, the set of $y_t$ with $t \in \mathbb{N}$ for discrete time systems (and $t \in [0, \infty)$ for the continuous-time case). These trajectories can be interpreted

as linear operators mapping from $\mathbb{R}^m$ (the space of observations $y$) into the time domain ($\mathbb{N}$ or $[0, \infty)$) and vice versa. The diagram below depicts this mapping:

$$X \longrightarrow \mathrm{Traj}(X) \longrightarrow C_q(\mathrm{Traj}(X))$$

Finally, $C_q(\mathrm{Traj}(X))$ is weighted in a suitable fashion by operators $S$ and $T$ and the trace is evaluated. This yields an element from the family of Binet-Cauchy kernels.

In the following we discuss several kernels and we show that they differ essentially in how the mapping into a dynamical system occurs (discrete-time or continuous time, fully observed or partial observations), whether any other preprocessing is carried out on $C_q(\mathrm{Traj}(X))$ (such as QR decomposition in the case of the kernel proposed by [10] and rediscovered by [17]), or which weighting $S, T$ is chosen.

### 3.2 Dynamical Systems Kernels

We begin with kernels on dynamical systems (12) as proposed in [14]. Set $S = \mathbf{1}$, $q = 1$ and $T$ to be the diagonal operator with entries $e^{-\lambda t}$. In this case the Binet-Cauchy kernel between systems $X$ and $X'$ becomes

$$\mathrm{tr}\, C_q(S\, \mathrm{Traj}(X)\, T\, \mathrm{Traj}(X')^\top) = \sum_{i=1}^{\infty} e^{-\lambda t} y_t^\top y_t'. \tag{13}$$

Since $y_t, y_t'$ are random variables, we also need to take expectations over $w_t, v_t, w_t', v_t'$. Some tedious yet straightforward algebra [14] allows us to compute (13) as follows:

$$k(X, X') = x_0^\top M_1 x_0' + \frac{1}{e^\lambda - 1}\, \mathrm{tr}\, [SM_2 + R], \tag{14}$$

where $M_1, M_2$ satisfy the Sylvester equations:

$$M_1 = e^{-\lambda} Q^\top P^\top P' Q' + e^{-\lambda} Q^\top M_1 Q' \text{ and } M_2 = P^\top P' + e^{-\lambda} Q^\top M_2 Q'. \tag{15}$$

Such kernels can be computed in $O(n^3)$ time [5]. Analogous expressions for continuous-time systems exist [14]. In Section 4 we will use this kernel to compute similarities between video sequences, after having encoded the latter as a dynamical system. This will allow us to compare sequences of different length, as they are all mapped to dynamical systems in the first place.

### 3.3 Martin Kernel

A characteristic property of (14) is that it takes initial conditions of the dynamical system into account. If this is not desired, one may choose to pick only the *subspace* spanned by the trajectory $y_t$. This is what was proposed in [10].[2]

More specifically, set $S = T = \mathbf{1}$, consider the trajectory upto only a finite number of time steps, say up to $n$, and let $q = n$. Furthermore let $\mathrm{Traj}(X) = Q_X R_X$ denote the QR-decomposition of $\mathrm{Traj}(X)$, where $Q_X$ is an orthogonal matrix and $R_X$ is upper triangular. Then it can be easily verified the kernel proposed by [10] can be written as

$$k(X, X') = \mathrm{tr}\, C_q(SQ_X TQ_{X'}^\top) = \det(Q_X Q_{X'}^\top). \tag{16}$$

This similarity measure has been used by Soatto, Doretto, and coworkers [4] for the analysis and computation of similarity in video sequences. Subsequently Wolf and Shashua [17] modified (16) to allow for kernels: to deal with determinants on a possibly infinite-dimensional feature space they simply project the trajectories on a reduced set of points in feature space.[3] This is what [17] refer to as a kernel on sets.

### 3.4 Graph Kernels

Yet another class of kernels can be seen to fall into this category: the graph kernels proposed in [6, 13, 9, 8]. Denote by $G(V, E)$ a graph with vertices $V$ and edges $E$. In some cases, such as in the analysis of molecules, the vertices will be equipped with labels $L$. For recovering these kernels from our framework we set $q = 1$ and systematically map graph kernels to dynamical systems.

We denote by $x_t$ a probability distribution over the set of vertices at time $t$. The time-evolution $x_t \rightarrow x_{t+1}$ occurs by performing a random walk on the graph $G(V, E)$. This yields $x_{t+1} = WD^{-1}x_t$, where $W$ is the connectivity matrix of the graph and $D$ is a diagonal matrix where $D_{ii}$ denotes the outdegree of vertex $i$. For continuous-time systems one uses $x(t) = \exp(-\tilde{L}t)x(0)$, where $\tilde{L}$ is the normalized graph Laplacian [9].

In the graph kernels of [9, 13] one assumes that the variables $x_t$ are directly observed and no special mapping is required in order to obtain $y_t$. Various choices of $S$ and $T$ yield the following kernels:

- [9] consider a snapshot of the diffusion process at $t = \tau$. This amounts to choosing $T = \mathbf{1}$ and a $S$ which is zero except for a diagonal entry at $\tau$.
- The inverse Graph-Laplacian kernel proposed in [13] uses a weighted combination of diffusion process and corresponds to $S = W$ a diagonal weight matrix.
- The model proposed in [6] can be seen as using a partially observable model: rather than observing the states directly, we only observe the labels emitted at the states. If we associate this mapping from states to labels with the matrix $P$ of (12), set $T = \mathbf{1}$ and let $S$ be the projector on the first $n$ time instances, we recover the kernels from [6].

So far, we deliberately made no distinction between kernels *on* graphs and kernels *between* graphs. This is for good reason: the trajectories depend on both *initial conditions* and the *dynamical system* itself. Consequently, whenever we want to consider kernels between initial conditions, we choose the same dynamical system in both cases. Conversely, whenever we want to consider kernels between dynamical systems, we average over initial conditions. This is what allows us to cover all the aforementioned kernels in one framework.

### 3.5 Extensions

Obviously the aforementioned kernels are just specific instances of what is possible by using kernels of Theorem 9. While it is pretty much impossible to enumerate all combinations, we give a list of suggestions for possible kernels below:

- Use the continuous-time diffusion process and a partially observable model. This would extend the diffusion kernels of [9] to comparisons between vertices of a labeled graph (e.g. atoms in a molecule).
- Use diffusion processes to define similarity measures between graphs.
- Compute the determinant of the trajectory associated with an $n$-step random walk on a graph, that is use $C_q$ with $q = n$ instead of $C_1$. This gives a kernel analogous to the one proposed by Wolf and Shashua [17], however without the whitening incurred by the QR factorization.
- Take Fredholm determinants of the above mentioned trajectories.
- Use a nonlinear version of the dynamical system as described in [14].

## 4 Experiments

To test the utility of our kernels we applied it to the task of clustering short video clips. We randomly sampled 480 short clips from the movie *Kill Bill* and model them as linear ARMA models (see Section 3.1).

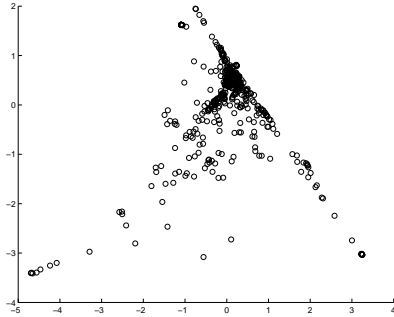

**Figure 1:** LLE embeddings of 480 random clips from Kill Bill

The sub-optimal procedure outlined in [4] was used for estimating the model parameters $P$, $Q$, $R$ and, $S$ and the kernels described in Section 3.2 were applied to these models. Locally Linear Embedding (LLE) [12] was used to cluster and embed the clips in two dimensions. The two dimensional embedding obtained by LLE is depicted in Figure 1. We randomly selected a few data points from Figure 1 and depict the first frame of the corresponding clips in Figure 2.

Observe the linear cluster (with a projecting arm) in Figure 1. This corresponds to clips which are temporally close to each other and hence have similar dynamics. For instance clips in the far right depict a person rolling in the snow while those in the far left corner depict a sword fight while clips in the center involve conversations between two characters. A naïve comparison of the intensity values or a dot product of the actual clips would not be able to extract such semantic information. Even though the camera angle varies with time our kernel is able to successfully pick out the underlying dynamics of the scene. These experiments are encouraging and future work will concentrate on applying this to video sequence querying.

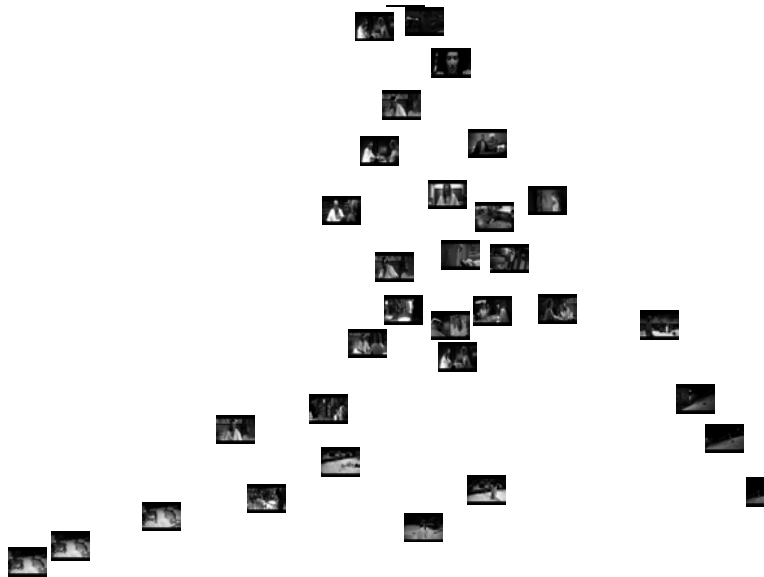

Figure 2: LLE embeddings of a subset of our dataset. A larger version is available from `http://mlg.anu.edu.au/~vishy/papers/KillBill.png`

## 5 Discussion

In this paper, we introduced a unifying framework for defining kernels on discrete objects using the Binet-Cauchy theorem on compounds of the Fredholm operators. We demonstrated that many of the previously known kernels can be explained neatly by our framework. In particular many graph kernels and dynamical system related kernels fall out as natural special cases. The main advantage of our unifying framework is that it allows kernel engineers to use domain knowledge in a principled way to design kernels for solving real life problems.

**Acknowledgement**   We thank Stephane Canu and René Vidal for useful discussions. National ICT Australia is supported by the Australian Government's Program Backing Australia's Ability. This work was partly supported by grants of the Australian Research Council. This work was supported by the IST Programme of the European Community, under the Pascal Network of Excellence, IST-2002-506778.

## Footnotes

[1]Eq. (9) can be seen as follows: the compound matrix of an orthogonal matrix is orthogonal and consequently its determinant is unity. Subsequently use an SVD factorization of the argument of the compound operator to compute the determinant of the compound matrix of a diagonal matrix.

[2]Martin [10] suggested the use of Cepstrum coefficients of a dynamical system to define a Euclidean metric. Later De Cock and Moor [2] showed that this distance is, indeed, given by the computation of subspace angles, which can be achieved by computing the QR-decomposition.

[3]To be precise, [17] are unaware of the work of [10] or of [2] and they rediscover the notion of subspace angles for the purpose of similarity measures.

## References

[1] C. J. C. Burges and V. Vapnik. A new method for constructing artificial neural networks. Interim technical report, ONR contract N00014 - 94-c-0186, AT&T Bell Laboratories, 1995.

[2] K. De Cock and B. De Moor. Subspace angles between ARMA models. *Systems and Control Letter*, 46:265 – 270, 2002.

[3] C. Cortes, P. Haffner, and M. Mohri. Rational kernels. In *Proceedings of Neural Information Processing Systems 2002*, 2002. in press.

[4] G. Doretto, A. Chiuso, Y.N. Wu, and S. Soatto. Dynamic textures. *International Journal of Computer Vision*, 51(2):91 – 109, 2003.

[5] J. D. Gardiner, A. L. Laub, J. J. Amato, and C. B. Moler. Solution of the Sylvester matrix equation $AXB^\top + CXD^\top = E$. *ACM Transactions on Mathematical Software*, 18(2):223 – 231, 1992.

[6] T. Gärtner, P.A. Flach, and S. Wrobel. On graph kernels: Hardness results and efficient alternatives. In B. Schölkopf and M. Warmuth, editors, *Sixteenth Annual Conference on Computational Learning Theory and Seventh Kernel Workshop, COLT*. Springer, 2003.

[7] W. Gröbner. *Matrizenrechnung*. BI Hochschultaschenbücher, 1965.

[8] H. Kashima, K. Tsuda, and A. Inokuchi. Marginalized kernels between labeled graphs. In *Proceedings of the $20^{\text{th}}$ International Conference on Machine Learning (ICML)*, Washington, DC, United States, 2003.

[9] I.R. Kondor and J. D. Lafferty. Diffusion kernels on graphs and other discrete structures. In *Proceedings of the ICML*, 2002.

[10] R.J. Martin. A metric for ARMA processes. *IEEE Transactions on Signal Processing*, 48(4):1164 – 1170, 2000.

[11] A. Pinkus. Spectral properties of totally positive kernels and matrices. In M. Gasca and C. A. Miccheli, editors, *Total Positivity and its Applications*, volume 359 of *Mathematics and its Applications*, pages 1–35. Kluwer, March 1996.

[12] S. Roweis and L. K. Saul. Nonlinear dimensionality reduction by locally linear embedding. *Science*, 290:2323 – 2326, December 2000.

[13] A.J. Smola and I.R. Kondor. Kernels and regularization on graphs. In B. Schölkopf and M. K. Warmuth, editors, *Proceedings of the Annual Conference on Computational Learning Theory*, Lecture Notes in Computer Science. Springer, 2003.

[14] A.J. Smola, R. Vidal, and S.V.N. Vishwanathan. Kernels and dynamical systems. *Automatica*, 2004. submitted.

[15] S.V.N. Vishwanathan and A.J. Smola. Fast kernels on strings and trees. In *Proceedings of Neural Information Processing Systems 2002*, 2002.

[16] J. C. Willems. From time series to linear system. I. Finite-dimensional linear time invariant systems. *Automatica J. IFAC*, 22(5):561 – 580, 1986.

[17] L. Wolf and A. Shashua. Learning over sets using kernel principal angles. *Jounal of Machine Learning Research*, 4:913 – 931, 2003.